# Semi-supervised Learning using Sparse Eigenfunction Bases

**Kaushik Sinha**
Dept. of Computer Science and Engineering
Ohio State University
Columbus, OH 43210
sinhak@cse.ohio-state.edu

**Mikhail Belkin**
Dept. of Computer Science and Engineering
Ohio State University
Columbus, OH 43210
mbelkin@cse.ohio-state.edu

## Abstract

We present a new framework for semi-supervised learning with sparse eigenfunction bases of kernel matrices. It turns out that when the data has clustered, that is, when the high density regions are sufficiently separated by low density valleys, each high density area corresponds to a unique representative eigenvector.

Linear combination of such eigenvectors (or, more precisely, of their Nystrom extensions) provide good candidates for good classification functions when the *cluster assumption* holds. By first choosing an appropriate basis of these eigenvectors from unlabeled data and then using labeled data with Lasso to select a classifier in the span of these eigenvectors, we obtain a classifier, which has a very sparse representation in this basis. Importantly, the sparsity corresponds naturally to the cluster assumption.

Experimental results on a number of real-world data-sets show that our method is competitive with the state of the art semi-supervised learning algorithms and outperforms the natural base-line algorithm (Lasso in the Kernel PCA basis).

## 1   Introduction

Semi-supervised learning, i.e., learning from both labeled and unlabeled data has received considerable attention in recent years due to its potential in reducing the need for expensive labeled data. However, to make effective use of unlabeled examples one needs to make some assumptions about the connection between the process generating the data and the process of assigning labels. There are two important assumptions popular in semi-supervised learning community the "cluster assumption" [CWS02] and the "manifold assumption" [BNS06] as well as a number of model-based methods, such as Naive Bayes [HTF03]. In particular, the cluster assumption can be interpreted as saying that two points are likely to have the same class labels if they can be connected by a path passing through a high density area. In other words two high density areas with different class labels must be separated by a low density valley.

In this paper, we develop a framework for semi-supervised learning when the cluster assumption holds. Specifically, we show that when the high density areas are sufficiently separated, a few appropriately chosen eigenfunctions of a convolution operator (which is the continuous counterpart of the kernel matrix) represents the high density areas reasonably well. Under the ideal conditions each high density area can be represented by a single unique eigenfunction called the "representative" eigenfunction. If the cluster assumption holds, each high density area will correspond to just one class label and thus a sparse linear combination of these representative eigenfunctions would be a good classifier. Moreover, the basis of such eigenfunctions can be learned using only the unlabeled data by constructing the Nystrom extension of the eigenvectors of an appropriate kernel matrix.

Thus, given unlabeled data we construct the basis of eigenfunctions and then apply $L^1$ penalized optimization procedure Lasso [Tib96] to fit a sparse linear combination of the basis elements to

the labeled data. We provide a detailed theoretical analysis of the algorithm and show that it is comparable to the state-of-the-art on several common UCI datasets.

The rest of the paper is organized as follows. In section 2 we provide the proposed framework for semi-supervised learning and describe the algorithm. In section 3 we provide an analysis of this algorithm to show that it can consistently identify the correct model. In section 4 we provide experimental results on synthetic and real datasets and finally we conclude with a discussion in section 5.

## 2 Semi-supervised Learning Framework

### 2.1 Outline of the Idea

In this section we present a framework for semi-supervised learning under the cluster assumption. Specifically we will assume that (i) data distribution has natural clusters separated by regions of low density and (ii) the label assignment conforms to these clusters.

The recent work of [SBY08a, SBY08b] shows that if the (unlabeled) data is clustered, then for each high density region there is a unique (representative) eigenfunction of a convolution operator, which takes positive values for points in the chosen cluster and whose values are close to zero everywhere else (no sign change). Moreover, it can be shown (e.g., [RBV08]) that these eigenfunctions can be approximated from the eigenvectors of a kernel matrix obtained from the unlabeled data.

Thus, if the cluster assumption holds we expect each cluster to have exactly one label assignment. Therefore eigenfunctions corresponding to these clusters should produce a natural sparse basis for constructing a classification function.

This suggests the following learning strategy:

1. From unlabeled and labeled data obtain the eigenvectors of the Gaussian kernel matrix.

2. From these eigenvectors select a subset of candidate eigenvectors without sign change.

3. Using the labeled data, apply Lasso (sparse linear regression) in the constructed basis to obtain a classifier.

4. Using the Nystrom extension (see [BPV03]), extend the eigenvectors to obtain the classification function defined everywhere.

**Connection to Kernel PCA ( [SSM98]).** We note that our method is related to KPCA, where data is projected onto the space spanned by the top few eigenvectors of the kernel matrix and classification or regression task can be performed in that projected space. The important difference is that we choose a subset of the eigenvectors in accordance to the cluster assumption. We note that the method simply using the KPCA basis does not seem to benefit from unlabeled data and, in fact, cannot outperform the standard fully supervised SVM classifier. On the other hand, our algorithm using a basis subselection procedure shows results comparable to the state of the art.

This is due to two reasons. We will see that each cluster in the data corresponds to its unique representative eigenvector of the kernel matrix. However, this eigenvector may not be among the top eigenvectors and may thus be omitted when applying KPCA. Alternatively, if the representative eigenvector is included, it will be included with a number of other uninformative eigenvectors resulting in poor performance due to overfitting.

We now proceed with the detailed discussion of our algorithm and its analysis.

### 2.2 Algorithm

The focus of our discussion will be binary classification in the semi-supervised setting. Given $l$ labeled examples $\{(\boldsymbol{x}_i, y_i)\}_{i=1}^{l}$ sampled from an underlying joint probability distribution $\mathcal{P}_{\mathcal{X},\mathcal{Y}}$, $\mathcal{X} \subset \mathbb{R}^d, \mathcal{Y} = \{-1, 1\}$, where $\boldsymbol{x}_i$s are the data points, $y_i$s are their corresponding labels and $u$ unlabeled examples $\{\boldsymbol{x}_i\}_{i=l+1}^{l+u}$ drawn iid from the marginal distribution $\mathcal{P}_{\mathcal{X}}$, we choose a Gaussian kernel $k(\boldsymbol{x}, \boldsymbol{z}) = \exp\left(-\frac{\|\boldsymbol{x}-\boldsymbol{z}\|^2}{2\omega^2}\right)$ with kernel bandwidth $\omega$ to construct the kernel matrix $K$ where $K_{ij} = \frac{1}{u}k(\boldsymbol{z}_i, \boldsymbol{z}_j)$. Let $(\lambda_i, \boldsymbol{v}_i)_{i=1}^{u}$ be the eigenvalue-eigenvector pair of $K$ sorted by the non-increasing eigenvalues. It has been shown ([SBY08a, SBY08b]) that when data distribution $\mathcal{P}_{\mathcal{X}}$

has clusters, for each high density region there is a unique representative eigenfunction of a convolution operator that takes positive values around the chosen cluster and is close to zero everywhere else. Moreover these eigenfunctions can be approximated from the eigenvectors of a kernel matrix obtained from the unlabeled data ([RBV08]), thus for each high density region there is a unique representative eigenvector of the kernel matrix that takes only positive or negative values in the chosen cluster and is nearly zero everywhere else (no sign change).

If the cluster assumption holds, i.e., each high density region corresponds to a portion of a pure class, then the classifier can be naturally expressed as a linear combination of the representative eigenfunctions. representative eigenvector basis and a linear combination of the representative eigenvectors will be a reasonable candidate for a good classification function. However, identifying representative eigenvectors is not very trivial because in real life depending on the separation between high density clusters the representative eigenvectors can have no sign change up to some small precision $\epsilon > 0$. Specifically, we say that a vector $\boldsymbol{e} = (e_1, e_2, ..., e_n) \in \mathbb{R}^n$ has no sign change up to precision $\epsilon$ if either $\forall i \ e_i > -\epsilon$ or $\forall i \ e_i < \epsilon$. Let $N_\epsilon$ be the set of indices of all eigenvectors that have no sign change up to precision $\epsilon$. If $\epsilon$ is chosen properly, $N_\epsilon$ will contain representative eigenvectors (note that the set $N_\epsilon$ and the set $\{1, 2, ..., |N_\epsilon|\}$ are not necessarily the same). Thus, instead of identifying the representative eigenvectors, we carefully select a small set containing the representative eigenvectors. Our goal is to learn a linear combination of the eigenvectors $\sum_{i \in N_\epsilon} \beta_i \boldsymbol{v}_i$ which minimizes classification error on the labeled examples and the coefficients corresponding to non-representative eigenvectors are zeros. Thus, the task is more of model selection or sparse approximation.

Standard approach to get a sparse solution is to minimize a convex loss function $V$ on the labeled examples and apply a $L^1$ penalty (on $\beta_i$s). If we select $V$ to be square loss function, we end up solving the $L^1$ penalized least square or so called Lasso [Tib96], whose consistency property was studied in [ZY06]. Thus we would seek a solution of the form

$$\arg\min_{\boldsymbol{\beta}} (\boldsymbol{y} - \Psi\boldsymbol{\beta})^T (\boldsymbol{y} - \Psi\boldsymbol{\beta}) + \lambda ||\beta||_{L^1} \tag{1}$$

which is a convex optimization problem, where $\Psi$ is the $l \times |N_\epsilon|$ design matrix whose $i^{th}$ column is the first $l$ elements of $\boldsymbol{v}_{N_\epsilon(i)}$, $\boldsymbol{y} \in \mathbb{R}^l$ is the label vector, $\boldsymbol{\beta}$ is the vector of coefficients and $\lambda$ is a regularization parameter. Note that solving the above problem is equivalent to solving

$$\arg\min_{\boldsymbol{\beta}} (\boldsymbol{y} - \Psi\boldsymbol{\beta})^T (\boldsymbol{y} - \Psi\boldsymbol{\beta}) \ \ s.t. \sum_{i \in N_\epsilon} |\beta_i| \leq t \tag{2}$$

because for any given $\lambda \in [0, \infty)$, there exists a $t \geq 0$ such that the two problems have the same solution, and vice versa [Tib96]. We will denote the solution of Equation 2, by $\hat{\boldsymbol{\beta}}$. To obtain a classification function which is defined everywhere, we use the Nystrom extension of the $i^{th}$ eigenvector defined as $\psi_i(\boldsymbol{x}) = \frac{1}{\lambda_i \sqrt{l+u}} \sum_{j=1}^{l+u} \boldsymbol{v}_i(\boldsymbol{x}_j) k(\boldsymbol{x}, \boldsymbol{x}_j)$. Let the set $\mathcal{T}$ contains indices of all nonzero $\hat{\beta}_i$s. Using Nystrom extension, classification function is given by, $f(\boldsymbol{x}) = \sum_{i \in \mathcal{T}} \hat{\beta}_i \psi_i(\boldsymbol{x}) = \sum_{i=1}^{l+u} \mathrm{w}_i k(\boldsymbol{x}_i, \boldsymbol{x})$, where, $\mathrm{w} \in \mathbb{R}^u$ is a weight vector whose $i^{th}$ element is given by

$$\mathrm{w}_i = \sum_{j \in \mathcal{T}} \frac{\hat{\beta}_j \boldsymbol{v}_j(\boldsymbol{x}_i)}{\lambda_j \sqrt{u}} \tag{3}$$

and can be computed while training.

---

Algorithm for Semi-supervised Learning

---

**Input:** $\{(\boldsymbol{x}_i, y_i)\}_{i=1}^l, \ \{\boldsymbol{x}_i\}_{i=l+1}^{l+u}$
**Parameters:** $\omega, t, \epsilon$

1. Construct kernel matrix $K$ from $l + u$ unlabeled examples $\{\boldsymbol{x}_i\}_{i=1}^{l+u}$.

2. Select set $N_\epsilon$ containing indices of the eigenvectors with no sign change up to precision $\epsilon$.

3. Construct design matrix $\Psi$ whose $i^{th}$ column is top $l$ rows of $\boldsymbol{v}_{N_\epsilon(i)}$.

4. Solve Equation 2 to get $\hat{\boldsymbol{\beta}}$ and calculate weight vector w using Equation 3.

5. Given a test point $\boldsymbol{x}$, predict its label as $y = sign\left(\sum_{i=1}^u k(\boldsymbol{x}_i, \boldsymbol{x}) \mathrm{w}_i\right)$

---

# 3 Analysis of the Algorithm

The main purpose of the analysis is, (i) to estimate the amount of separation required among the high density regions which ensures that each high density region can be well represented by a unique (representative) eigenfunction, (ii) to estimate the number of unlabeled examples required so that eigenvectors of kernel matrix can approximate the eigenfunctions of a convolution operator (defined below) and (iii) to show that using few labeled examples Lasso can consistently identify the correct model consisting of linear combination of representative eigenvectors.

Before starting the actual analysis, we first note that the continuous counterpart of the Gram matrix is a convolution operator $L_K : L^2(\mathcal{X}, \mathcal{P}_\mathcal{X}) \to L^2(\mathcal{X}, \mathcal{P}_\mathcal{X})$ defined by,

$$(L_K f)(\boldsymbol{x}) = \int_\mathcal{X} k(\boldsymbol{x}, \boldsymbol{z}) f(\boldsymbol{z}) d\mathcal{P}_\mathcal{X}(\boldsymbol{z}) \tag{4}$$

The eigenfunctions of the symmetric positive definite operator $L_K$ will be denoted by $\phi_i^L$.

Next, we briefly discuss the effectiveness of model selection using Lasso (established by [ZY06]) which will be required for our analysis. Let $\hat{\boldsymbol{\beta}}_l(\lambda)$ be the solution of Equation 1 for a chosen regularization parameter $\lambda$. In [ZY06] a concept of *sign consistency* was introduced which states that Lasso is sign consistent if, as $l$ tends to infinity, signs of $\hat{\boldsymbol{\beta}}_l(\lambda)$ matches with the signs of $\boldsymbol{\beta}^*$ with probability 1, where $\boldsymbol{\beta}^*$ is the coefficients of the correct model. Note that since we are expecting a sparse model, matching zeros of $\hat{\boldsymbol{\beta}}_l(\lambda)$ to the zeros of $\boldsymbol{\beta}^*$ is not enough, but in addition, matching the signs of the non zero coefficients ensures that the true model will be selected. Next, without loss of generality assume $\beta^* = (\beta_1^*, \cdots, \beta_q^*, \beta_{q+1}^*, \cdots, \beta_{|N_\epsilon|}^*)$ has only first $q$ terms non-zero, i.e., only $q$ predictors describe the model and rest of the predictors are irrelevant in describing the model. Now let us write the first $q$ and $|N_\epsilon| - q$ columns of $\Psi$ as $\Psi_{(1)}$ and $\Psi_{(2)}$ respectively. Let $C = \frac{1}{l}\Psi^T\Psi$.

Note that, for a random design matrix, sign consistency is equivalent to *irrepresentable* condition (see [ZY06]). When $\boldsymbol{\beta}^*$ is unknown, in order to ensure that irrepresentable condition holds for all possible signs, it requires that $L^1$ norm of the regression coefficients corresponding to the irrelevant predictors to be less than 1, which can be written as $\mu_\Psi = \max_{\psi_j^u \in \Psi_{(2)}} \left\| \left(\Psi_{(1)}^T \Psi_{(1)}\right)^T \Psi_{(1)}^T \psi_j^u \right\|_1 <$ 1. The requirement $\mu_\Psi < 1$ is not new and have also appeared in the context of noisy or noiseless sparse recovery of signal [Tro04, Wai06, Zha08]. Note that Lasso is sign consistent if irrepresentable condition holds and the sufficient condition needed for irrepresentable condition to hold is given by the following result,

**Theorem 3.1.** *[ZY06] Suppose $\boldsymbol{\beta}^*$ has $q$ nonzero entries. Let the matrix $C'$ be normalized version of $C$ such that $C'_{ij} = \frac{C_{ij}}{C_{ii}}$ and $\max_{i,j,i\neq j} |C'_{ij}| \leq \frac{c}{2q-1}$ for a constant $0 \leq c < 1$, then strong irrepresentable condition holds.*

Our main result in the following shows that this sufficient condition is satisfied with high probability requiring relatively few labeled examples, as a result the correct model is identified consistently, which in turn describes a good classification function.

**Theorem 3.2.** *Let $q$ be the minimum number of columns of the design matrix $\Psi \in \mathbb{R}^{l \times |N_\epsilon|}$, constructed from $l$ labeled examples, that describes the sparse model. Then for any $0 < \delta < 1$, if the number of unlabeled examples $u$ satisfies $u > \frac{2048q^2 \log\left(\frac{2}{\delta}\right)}{g_{N_{\max}}^2 \lambda_{N_{\max}}^2}$, then with probability greater than $1 - \frac{\delta}{2} - 4\exp\left(-\frac{l\lambda_{N_{\max}}^2}{50q^2}\right)$, $\max_{i\neq j} |C'_{ij}| < \frac{1}{2q-1}$.*

where $\lambda_{N_{\max}}$ is the $N_{\max}^{th}$ (to be defined later) largest eigenvalue of $L_K$ and $g_{N_{\max}}$ is the $N_{\max}^{th}$ eigengap. Note that in our framework, unlabeled examples help polynomially fast in estimating the eigenfunctions while labeled examples help exponentially fast in identifying the sparse model consisting of representative eigenfunctions. Interestingly, in semi-supervised learning setting, similar role of labeled and unlabeled examples (in reducing classification error) has been reported in literature [CC96, RV95, SB07, SNZ08].

## 3.1 Brief Overview of the Analysis

As a first step of our analysis, in section 3.2, we estimate the separation requirement among the high density regions which ensures that each high density region (class) can be well represented by a unique eigenfunction. This allows us to express the classification task in this eigenfunction

basis where we look for a classification function consisting of linear combination of representative eigenfunctions only and thus relate the problem to sparse approximation from the model selection point of view, which is a well studied field [Wai06, ZH06, CP07].

As a second step in section 3.3, using perturbation results from [RBV08], we estimate the number of unlabeled examples required to ensure that Nystrom extensions of eigenvectors of $K$ approximate the eigenfunctions of the convolution operator $L_K$ reasonably well with high probability.

Finally, as a third step in section 3.4, we establish a concentration inequality, which along with result from the second step 2, ensures that as more and more labeled examples are used to fit the eigenfunctions basis to the data, the probability that Lasso identifies correct model consisting of representative eigenfunctions increases exponentially fast.

## 3.2 Separation Requirement

To motivate our discussion we consider binary classification problem where the marginal density can be considered as a mixture model where each class has its own probability density function, $p_1(\boldsymbol{x}), p_2(\boldsymbol{x})$ and corresponding mixing weights $\pi_1, \pi_2$ respectively. Thus, the density of the mixture is $p(\boldsymbol{x}) = \pi_1 p_1(\boldsymbol{x}) + \pi_2 p_2(\boldsymbol{x})$. We will use the following results from [SBY08a] specifying the behavior of the eigenfunction of $L_K$ corresponding to the largest eigenvalue.

**Theorem 3.3.** *[SBY08a] The top eigenfunction $\phi_0^L(\boldsymbol{x})$ of $L_K$ corresponding to the largest eigenvalue $\lambda_0$, (1) is the only eigenfunction with no sign change, (2) has multiplicity one, (3) is non zero on the support of the underlying density, (4) satisfies $|\phi_0^L(\boldsymbol{x})| \leq \frac{1}{\lambda_0}\sqrt{\int k^2(\boldsymbol{x},\boldsymbol{z})p(\boldsymbol{z})d\boldsymbol{z}}$ (Tail decay property), where $p$ is the underlying probability density function.*

Note that the last (tail decay) property above is not restricted to the top eigenfunction alone but is satisfied by all eigenfunctions of $L_K$. Now, consider applying $L_K$ to the three cases when the underlying probability distributions are $p_1, p_2$ and $p$. The largest eigenvalues and corresponding eigenfunctions in the above three cases are $\lambda_0^1, \lambda_0^2, \lambda_0$ and $\phi_0^{L,1}, \phi_0^{L,2}, \phi_0^L$ respectively. To show explicit dependency on the underlying probability distribution, we will denote the corresponding operators as $L_K^{p_1}, L_K^{p_2}$ and $L_K^p$ respectively. Clearly, $L_K^p = \pi_1 L_K^{p_1} + \pi_2 L_K^{p_2}$. Then we can write, $L_K^p \phi_0^{L,1}(\boldsymbol{x}) = \int k(\boldsymbol{x},\boldsymbol{z})\phi_0^{L,1}(\boldsymbol{z})p(\boldsymbol{z})d\boldsymbol{z} = \pi_1 \lambda_0^1 \left(\phi_0^{L,1} + T_1(\boldsymbol{x})\right)$ where, $T_1(\boldsymbol{x}) = \frac{\pi_2}{\pi_1 \lambda_0^1}\int k(\boldsymbol{x},\boldsymbol{z})\phi_0^{L,1}(\boldsymbol{z})p_2(\boldsymbol{z})d\boldsymbol{z}$. In a similar way we can write, $L_K^p \phi_0^{L,2}(\boldsymbol{x}) = \pi_2 \lambda_0^2 \left(\phi_0^{L,2} + T_2(\boldsymbol{x})\right)$ where, $T_2(\boldsymbol{x}) = \frac{\pi_1}{\pi_2 \lambda_0^2}\int k(\boldsymbol{x},\boldsymbol{z})\phi_0^{L,2}(\boldsymbol{z})p_1(\boldsymbol{z})d\boldsymbol{z}$. Thus, when $T_1(\boldsymbol{x})$ and $T_2(\boldsymbol{x})$ are small enough then $\phi_0^{L,1}$ and $\phi_0^{L,2}$ are eigenfunctions of $L_K^p$ with corresponding eigenvalues $\pi_1 \lambda_0^1$ and $\pi_2 \lambda_0^2$ respectively. Note that "separation condition" requirement refers to $T_1(\boldsymbol{x})$, $T_2(\boldsymbol{x})$ being small, so that eigenfunctions corresponding to the largest eigenvalues of convolution operator when applied to individual high density bumps are preserved in the case when convolution operator is applied to the mixture. Clearly, we can not expect $T_1(\boldsymbol{x})$, $T_2(\boldsymbol{x})$ to arbitrarily small if there is sufficient overlap between $p_1$ and $p_2$. Thus, we will restrict ourselves to the following class of probability distributions for each individual class which has reasonably fast tail decay.

**Assumption 1.** *For any $1/2 < \eta < 1$, let $\mathbb{M}(\eta, \mathcal{R})$ be the class of probability distributions such that its density function $p$ satisfies*
*1) $\int_{\mathcal{R}} p(\boldsymbol{x})d(\boldsymbol{x}) = \eta$ where $\mathcal{R}$ is the minimum volume ball around the mean of the distribution.*
*2) For any positive $t > 0$, smaller than the radius of $\mathcal{R}$, and for any point $\boldsymbol{z} \in \mathcal{X} \setminus \mathcal{R}$ with $dist(\boldsymbol{z}, \mathcal{R}) \geq t$, the volume $S = \{\boldsymbol{x} \in (\mathcal{X} \setminus \mathcal{R}) \cap B(\boldsymbol{z}, 3t/\sqrt{2})\}$ has total probability mass $\int_S p(\boldsymbol{x})d\boldsymbol{x} \leq C_1 \eta \exp\left(-\frac{dist^2(\boldsymbol{z},\mathcal{R})}{t^2}\right)$ for some $C_1 > 0$.*

where the distance between a point $\boldsymbol{x}$ and set $\mathcal{D}$ is defined as $dist(\boldsymbol{x}, \mathcal{D}) = \inf_{\boldsymbol{y} \in \mathcal{D}} ||\boldsymbol{x} - \boldsymbol{y}||$. With a little abuse of notation we will use $p \in \mathbb{M}(\eta, \mathcal{R})$ to mean that $p$ is the probability density function of a member of $\mathbb{M}(\eta, \mathcal{R})$. Now a rough estimate of separation requirement can be given by the following lemma.

**Lemma 3.1.** *Let $p_1 \in \mathbb{M}(\eta, \mathcal{R}_1)$ and $p_2 \in \mathbb{M}(\eta, \mathcal{R}_2)$ and let the minimum distance between $\mathcal{R}_1, \mathcal{R}_2$ be $\Delta$. If $\Delta = \Omega^*\left(\omega\sqrt{d}\right)$ then $T_1(\boldsymbol{x})$ and $T_2(\boldsymbol{x})$ can be made arbitrarily small for all $\boldsymbol{x} \in \mathcal{X}$.*

The estimate of $\Delta$ in the above lemma, where we hide the log factor by $\Omega^*$, is by no means tight, nevertheless, it shows that separation requirement refers to existence of a low density valley between

two high density regions each corresponding to one of the classes. This separation requirement is roughly of the same order required to learn mixture of Gaussians [Das99]. Note that, provided separation requirement is satisfied, $\phi_0^{L,1}$ and $\phi_0^{L,2}$ are not necessarily the top two eigenfunctions of $L_K$ corresponding to the two largest eigenvalues but can be quite far down the spectrum of $L_K^p$ depending on the mixing weights $\pi_1, \pi_2$. Next, the following lemma suggests that we can say more about the eigenfunction corresponding to the largest eigenvalue.

**Lemma 3.2.** *For any $\frac{e}{1+e} < \eta < 1$, let $q \in \mathbb{M}(\eta, \mathcal{R})$. If $\phi_0^L$ is the eigenfunction of $L_K^q$ corresponding to the largest eigenvalue $\lambda_0$ then there exists a $C_1 > 0$ such that*

*1) For all $\boldsymbol{x} \in \mathcal{X} \setminus \mathcal{R}, |\phi_0^L(\boldsymbol{x})| \leq \frac{\sqrt{(C_1+\eta)}}{\lambda_0} \exp\left(-\frac{dist^2(\boldsymbol{x}, \mathcal{R})}{2\omega^2}\right)$*
*2) For all $\boldsymbol{z} \in \mathcal{R}$ and $\boldsymbol{x} \in \mathcal{X} \setminus \mathcal{R}, |\phi_0^L(\boldsymbol{z})| \geq |\phi_0^L(\boldsymbol{x})|$*

Thus for each class, top eigenfunction corresponding to the largest eigenvalue represents high density region reasonably well, outside high density region is has lower absolute value and decays exponentially fast.

### 3.3 Finite Sample Results
We start with the following assumption.

**Assumption 2.** *The $N_{\max}$ largest eigenvalues of $L_K$ and $K$, where $N_{\max} = \max_i\{i : i \in N_\epsilon\}$, are simple and bounded away from zero.*

Note that Nystrom extension $\psi_i$s are eigenfunctions of an operator $L_{K,\mathcal{H}} : \mathcal{H} \to \mathcal{H}$, where $\mathcal{H}$ is the unique RKHS defined by the chosen Gaussian kernel and all the eigenvalues of $K$ are also eigenvalues of $L_{K,\mathcal{H}}$ ([RBV08]). There are two implications of Assumption 2. The first one is due to the *bounded away from zero* part, which ensures that if we restrict to $\psi_i \in \mathcal{H}$ corresponding to the largest $N_{\max}$ eigenvalues, then each of them is square integrable hence belongs to $L^2(\mathcal{X}, \mathcal{P}_\mathcal{X})$. The second implication due to the *simple* part, ensures that eigenfunctions corresponding to the $N_{\max}$ largest eigenvalues are uniquely defined and so are the orthogonal projections on to them. Note that if any eigenvalue has multiplicity greater than one then the corresponding eigenspace is well defined but not the individual eigenfunctions. Thus, Assumption 2 enables us to compare how close each $\psi_i$ is to some other function in $L^2(\mathcal{X}, \mathcal{P}_\mathcal{X})$ in $L^2(\mathcal{X}, \mathcal{P}_\mathcal{X})$ norm sense. Let $g_{N_{\max}}$ be the $N_{\max}^{th}$ eigengap when eigenvalues of $L_K$ are sorted in non increasing order. Then we have the following results.

**Lemma 3.3.** *Suppose Assumption 2 holds and the top $N_{\max}$ eigenvalues of $L_K$ and $K$ are sorted in the decreasing order. Then for any $0 < \delta < 1$ and for any $i \in N_\epsilon$, with probability at least $(1 - \delta)$,*

$$\|\psi_i - \phi_i^L\|_{L^2(\mathcal{X}, \mathcal{P}_\mathcal{X})} = \frac{2}{g_{N_{\max}}}\sqrt{\frac{2\log(2/\delta)}{u\lambda_i}}$$

**Corollary 3.1.** *Under the above conditions, for any $0 < \delta < 1$ and for any $i, j \in N_\epsilon$, with probability at least $(1 - \delta)$ the following holds,*

*1) $\left|\langle\psi_i, \psi_j\rangle_{L^2(\mathcal{X}, \mathcal{P}_\mathcal{X})}\right| \leq \left(\frac{8\log(2/\delta)}{g_{N_{\max}}^2\sqrt{\lambda_i\lambda_j}}\right)\frac{1}{u} + \left(\frac{\sqrt{8\log(2/\delta)}}{g_{N_{\max}}}\left(\frac{1}{\sqrt{\lambda_i}} + \frac{1}{\sqrt{\lambda_j}}\right)\right)\frac{1}{\sqrt{u}}$*
*2) $1 - \left(\sqrt{\frac{8\log(2/\delta)}{g_{N_{\max}}^2\lambda_i}}\right)\frac{1}{\sqrt{u}} \leq \|\psi_i^u\|_{L^2(\mathcal{X}, \mathcal{P}_\mathcal{X})} \leq 1 + \left(\sqrt{\frac{8\log(2/\delta)}{g_{N_{\max}}^2\lambda_i}}\right)\frac{1}{\sqrt{u}}$*

### 3.4 Concentration Results
Having established that $\{\psi_i\}_{i \in N_\epsilon}$ approximate the top $N_\epsilon$ eigenfunctions of $L_K$ reasonably well, next, we need to consider what happens when we restrict each of the $\psi_i$s to finite labeled examples. Note that the design matrix $\Psi \in \mathbb{R}^{l \times |N_\epsilon|}$ is constructed by restricting the $\{\psi_j\}_{j \in N_\epsilon}$ to $l$ labeled data points $\{\boldsymbol{x}_i\}_{i=1}^l$ such that the $i^{th}$ column of $\Psi$ is $(\psi_{N_\epsilon(i)}(\boldsymbol{x}_1), \psi_{N_\epsilon(i)}(\boldsymbol{x}_2), \cdots, \psi_{N_\epsilon(i)}(\boldsymbol{x}_l))^T \in \mathbb{R}^l$. Now consider the $|N_\epsilon| \times |N_\epsilon|$ matrix $C = \frac{1}{l}\Psi^T\Psi$ where, $C_{ij} = \frac{1}{l}\sum_{k=1}^l \psi_{N_\epsilon(i)}(\boldsymbol{x}_k)\psi_{N_\epsilon(j)}(\boldsymbol{x}_k)$. First, applying Hoeffding's inequality we establish,

**Lemma 3.4.** *For all $i, j \in N_\epsilon$ and $\epsilon_1 > 0$ the following two facts hold.*
$$\mathbb{P}\left(\left|\frac{1}{l}\sum_{k=1}^l[\psi_i(\boldsymbol{x}_k)]^2 - \mathbb{E}\left([\psi_i(X)]^2\right)\right| \geq \epsilon_1\right) \leq 2\exp\left(-\frac{l\epsilon_1^2\lambda_i^2}{2}\right)$$
$$\mathbb{P}\left(\left|\frac{1}{l}\sum_{k=1}^l\psi_i(\boldsymbol{x}_k)\psi_j(\boldsymbol{x}_k) - \mathbb{E}\left(\psi_i(X)\psi_j(X)\right)\right| \geq \epsilon_1\right) \leq 2\exp\left(-\frac{l\epsilon_1^2\lambda_i\lambda_j}{2}\right)$$

Next, consider the $|N_\epsilon| \times |N_\epsilon|$ normalized matrix $C'$ where $C'_{ij} = \frac{C_{ij}}{C_{ii}}$ and $C'_{ii} = 1$. To ensure that Lasso will consistently choose the correct model we need to show (see Theorem 3.1) that

$\max_{i \neq j} |C'_{ij}| < \frac{1}{2q-1}$ with high probability. Applying the above concentration result and finite sample results yields Theorem 3.2.

## 4  Experimental Results

### 4.1  Toy Dataset

Here we present a synthetic example in 2-D. Consider a binary classification problem where the positive examples are generated from a Gaussian distribution with mean $(0,0)$ and covariance matrix $[2\ 0;\ 0\ 2]$ and the negative examples are generated from a mixture of Gaussians having means and covariance matrices $(5,5), [2\ 1;\ 1\ 2]$ and $(7,7), [1.5\ 0;\ 0\ 1.5]$ respectively. The corresponding mixing weights are $0.4, 0.3$ and $0.3$ respectively. Left panel in Figure 1 shows the probability density of the mixture in blue and representative eigenfunctions of each class in green and magenta respectively using 1000 examples (positive and negative) drawn from this mixture. It is clear that each representative eigenfunction represents high density area of a particular class reasonably well. So intuitively a linear combination of them will represent a good decision function. In fact, the right panel of Fig 1 shows the regularization path for $L^1$ penalized least square regression with 20 labeled examples. The bold green and magenta lines shows the coefficient values for the representative eigenfunctions for different values of regularization parameter $t$. As can be seen, regularization parameter $t$ can be so chosen that the decision function will consist of a linear combination of representative eigenfunctions only. Note that these representative eigenfunctions need not be the top two eigenfunctions corresponding to the largest eigenvalues.

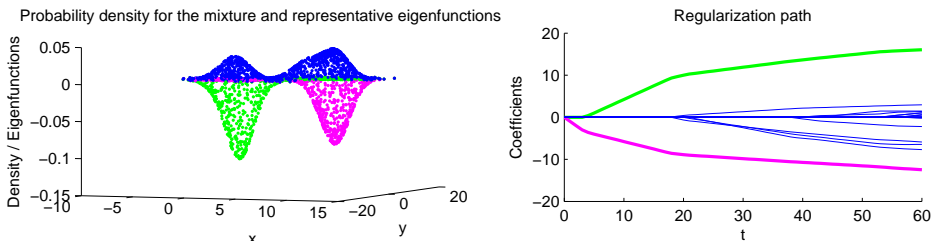

Figure 1: **Left panel:** Probability density of the mixture in blue and representative eigenfunctions in green and magenta. **Right panel:** Regularization path. Bold lines correspond to regularization path associated with representative eigenfunctions.

### 4.2  UCI Datasets

In this set of experiment we tested the effectiveness of our algorithm (we call it SSL_SEB) on some common UCI datasets. We compared our algorithm with state of the art semi-supervised learning (manifold regularization) method Laplacian SVM (LapSVM) [BNS06], fully supervised SVM and also two other kernel sparse regression methods. In KPCA+$L^1$ we selected top $|N_\epsilon|$ eigenvectors, and applied $L^1$ regularization, in KPCA_F+$L^1$ we selected the top 20 (fixed) eigenvectors of $K_u$ and applied $L^1$ regularization[1], where as in KPCA_max+$L^1$ we selected top $max$ eigenvectors, and applied $L^1$ regularization, where $max$ is the maximum index of set of eigenvectors in $N_\epsilon$, that is the index of the lowest eigenvector, chosen by our method. For both SVM and LapSVM we used RBF kernel. In each experiment a specified number of examples ($l$) were randomly chosen and labeled and the rest ($u$) were treated as unlabeled test set. Such random splitting was performed 30 times and the average is reported.

The results are reported in Table 1. As can be seen, for small number of labeled examples our method convincingly outperform SVM and is comparable to LapSVM. The result also suggests that instead of selecting top few eigenvectors, as is normally done in KPCA, selecting them by our method and then applying $L^1$ regularization yields better result. In particular, in case of IONOSPHERE and BREAST-CANCER data sets top $|N_\epsilon|$ (5 and 3 respectively) eigenvectors do not contain the representative ones. As a result in these two cases KPCA+$L^1$ performs very poorly. Table 2 shows that the solution obtained by our method is very sparse, where average sparsity is the average number of non-zero coefficients.

We note that our method does not work equally well for all datasets, and has generally higher variability than LapSVM.

| DATA SET | IONOSPHERE d=33, l+u=351 | | | HEART d=13, l+u=303 | | | WINE d=13, l+u=178 | | BREAST-CANCER d=30, l+u=569 | | VOTING d=16, l+u=435 | |
|---|---|---|---|---|---|---|---|---|---|---|---|---|
| # Labeled Data | l=10 | l=20 | l=30 | l=10 | l=20 | l=30 | l=10 | l=20 | l=5 | l=10 | l=10 | l=15 |
| SSL_SEB | **78.26** ±13.56 | **85.84** ±10.61 | **87.25** ±4.16 | **75.45** ±6.14 | **77.34** ±6.04 | **79.92** ±1.18 | 93.01 ±8.49 | **98.95** ±8.49 | 96.68 ±3.43 | 98.66 ±2.86 | 86.85 ±6.21 | 87.84 ±3.82 |
| KPCA+$L^1$ | 65.15 ±8.82 | 65.66 ±9.81 | 69.57 ±9.89 | 66.82 ±7.94 | 70.36 ±8.41 | 75.16 ±6.68 | 93.47 ±10.06 | 98.75 ±3.89 | 70.26 ±14.43 | 73.95 ±13.68 | 86.85 ±6.21 | 87.84 ±3.82 |
| KPCA_F+$L^1$ | 64.92 ±10.13 | 67.43 ±11.68 | 69.43 ±11.26 | 60.91 ±7.33 | 67.32 ±7.01 | 71.46 ±5.91 | 79.82 ±10.29 | 87.32 ±8.56 | 63.04 ±12.29 | 81.44 ±13.12 | 71.78 ±12.65 | 77.38 ±10.43 |
| KPC_max+$L^1$ | 59.76 ±10.23 | 64.73 ±11.62 | 66.89 ±12.45 | 57.26 ±5.16 | 60.16 ±6.69 | 63.36 ±6.15 | 84.62 ±9.63 | 89.96 ±9.26 | 59.32 ±15.18 | 73.95 ±8.97 | 71.78 ±12.65 | 77.38 ±10.43 |
| SVM | 65.16 ±10.87 | 72.09 ±10.04 | 79.8 ±9.94 | 64.61 ±11.63 | 73.16 ±5.95 | 76.55 ±4.29 | 83.98 ±10.25 | 88.12 ±11.68 | 72.83 ±17.56 | 97.32 ±8.65 | 81.53 ±16.05 | 88.51 ±5.88 |
| LapSVM | 71.17 ±7.33 | 77.18 ±4.07 | 81.32 ±3.81 | 74.91 ±5.55 | 75.33 ±6.08 | 77.43 ±3.14 | **98.33** ±5.33 | 97.67 ±1.57 | **98.95** ±2.32 | **99.72** ±1.42 | **89.52** ±1.43 | **89.97** ±1.26 |

Table 1: Classification Accuracies for different UCI datasets

| DATA SET | IONOSPHERE | HEART | WINE | BREAST-CANCER | VOTING |
|---|---|---|---|---|---|
| **SSL_SEB** | 2.83 / 5 | 4.63 / 9 | 3.52 / 6 | 2.10 / 3 | 2.02/ 3 |
| KPCA+$L^1$ | 3.23 / 5 | 5.84 / 9 | 3.8 / 6 | 2.78 / 3 | 2.02/ 3 |
| KPCA_F+$L^1$ | 6.05 / 20 | 8.11 / 20 | 6.12 / 20 | 4.70 / 20 | 3.05 / 20 |
| KPC_max+$L^1$ | 6.85 / 23 | 16.42 / 78 | 6.07 /16 | 10.81 / 57 | 2.02/ 3 |

Table 2: Average sparsity of our method for different UCI datasets. The notation $A/B$ represents average sparsity $A$ and number of eigenvectors ($|N_\epsilon|$ or 20).

### 4.3   Handwritten Digit Recognition

In this set of experiments we applied our method to the 45 binary classification problems that arise in pairwise classification of handwritten digits and compare its performance with LapSVM. For each pairwise classification problem, in each trial, 500 images of each digit in the USPS training set were chosen uniformly at random out of which 20 images were labeled and the rest were set aside for testing. This trial was repeated 10 times. For the LapSVM we set the regularization terms and the kernel as reported by [BNS06] for a similar set of experiments, namely we set $\gamma_A l = 0.005, \frac{\gamma_I l}{(u+l)^2} = 0.045$ and chose a polynomial kernel of degree 3. The results are shown[2] in Figure2. As can be seen our method is comparable to LapSVM.

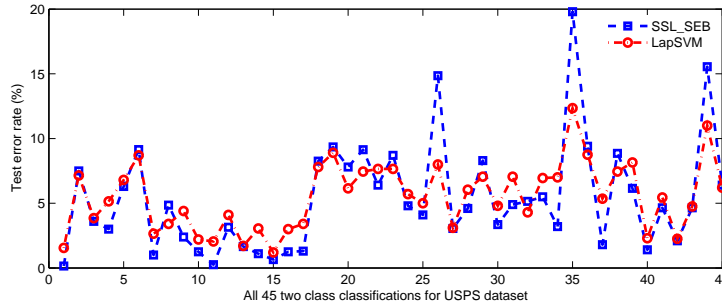

Figure 2: Classification results for USPS dataset

We also performed multi-class classification on USPS dataset. In particular, we chose all the images of digits 3, 4 and 5 from USPS training data set (there were 1866 in total) and randomly labeled 10 images from each class. Rest of the 1836 images were set aside for testing. Average prediction accuracy of LapSVM, after repeating this procedure 20 times, was $90.14\%$ as compared to $87.53\%$ of our method.

## 5   Conclusion

In this paper we have presented a framework for spectral semi-supervised learning based on the cluster assumption. We showed that the cluster assumption is equivalent to the classifier being sparse in a certain appropriately chosen basis and demonstrated how such basis can be computed using only unlabeled data. We have provided theoretical analysis of the resulting algorithm and given experimental results demonstrating that the resulting algorithm has performance comparable to the state-of-the-art for a number of data sets and dramatically outperforms the natural baseline of KPCA + Lasso.

## Footnotes

[1]We also selected 100 top eigenvectors and applied $L^1$ penalty but it gave worse result.

[2]It turned out that the cases where our method performed very poorly, the respective distances between the means of corresponding two classes were very small.

# References

[BNS06]   M. Belkin, P. Niyogi, and V. Sindhwani. Manifold Regularization: A Geometric Framework for Learning from Labeled and Unlabeled Examples. *Journal of Machine Learning Research*, 7:2399–2434, 2006.

[BPV03]   Y. Bengio, J-F. Paiement, and P. Vincent. Out-of-sample Extensions for LLE, Isomap, MDS, Eigenmaps and Spectral Clustering. In *NIPS*. 2003.

[CC96]    V. Castelli and T. M. Cover. The Relative Value of Labeled and Unlabeled Samples in Pattern Recognition with Unknown Mixing Parameters. *IEEE Transactions on Information Theory*, 42(6):2102–2117, 1996.

[CP07]    E. J. Candes and Y. Plan. Near Ideal Model Selection by $\ell_1$ Minimization, eprint arxiv:0801.0345. 2007.

[CWS02]   O. Chapelle, J. Weston, and B. Scholkopf. Cluster Kernels for Semi-supervised Learning. In *NIPS*. 2002.

[Das99]   S. Dasgupta. Learning Mixture of Gaussians. In *40th Annual Symposium on Foundations of Computer Science*, 1999.

[HTF03]   T. Hastie, R. Tibshirani, and J. Friedman. *The Elements of Statistical Learning Data Mining, Inference and Prediction*. Springer, 2003.

[RBV08]   L. Rosasco, M. Belkin, and E. De Vito. Perturbation Results for Learning Empirical Opertors. Technical Report TR-2008-052, Massachusetts Institute of Technology, Cambridge, MA, August 2008.

[RV95]    J. Ratsaby and S. Venkatesh. Learning From a Mixture of Labeled and Unlabeled Examples with Parametric Side Information. In *COLT*. 1995.

[SB07]    K. Sinha and M. Belkin. The Value of Labeled and Unlabeled Examples when the Model is Imperfect. In *NIPS*. 2007.

[SBY08a]  T. Shi, M. Belkin, and B. Yu. Data Spectroscopy: Eigenspace of Convolution Operators and Clustering. Technical report, Dept. of Statistics, Ohio State University, 2008.

[SBY08b]  T. Shi, M. Belkin, and B. Yu. Data Spectroscopy: Learning Mixture Models using Eigenspaces of Convolution Operators. In *ICML*. 2008.

[SNZ08]   A. Singh, R. D. Nowak, and X. Zhu. Unlabeled Data: Now it Helps Now it Doesn't. In *NIPS*. 2008.

[SSM98]   Bernhard Scholkopf, A. Smola, and Klaus-Robert Muller. Nonlinear Component Analysis as a Kernel Eigenvalue Problem. *Neural Computation*, 10:1299–1319, 1998.

[Tib96]   R. Tibshirani. Regression Shrinkage and Selection via the Lasso. *Journal of the Royal Statistical Society, Series B*, 58:267–288, 1996.

[Tro04]   J. A. Tropp. Greed is Good: Algorithmic Result for Sparse Approximation. *IEEE Trans. Info. Theory*, 50(10):2231–2242, 2004.

[Wai06]   M. Wainwright. Sharp Thresholds for Noisy and High-dimensional Recovery of Sparsity using $\ell_1$-constrained Quadratic Programming. Technical Report TR-709, Dept. of Statistics, U. C. Berkeley, September 2006.

[ZH06]    C. Zhang and J. Huang. Model Selection Consistency of Lasso in High Dimensional Linear Regression. Technical report, Dept. of Statistics, Rutgers University, 2006.

[Zha08]   T. Zhang. On consistency of feature selection using greedy least square regression. *Journal of Machine Learning Research*, 2008.

[ZY06]    P. Zhao and B. Yu. On Model Selection Consistency of Lasso. *Journal of Machine Learning Research*, 7:2541–2563, 2006.

